# COMPUTING MOTION USING RESISTIVE NETWORKS

Christof Koch, Jin Luo, Carver Mead
California Institute of Technology, 216-76, Pasadena, Ca. 91125

James Hutchinson
Jet Propulsion Laboratory, California Institute of Technology
Pasadena, Ca. 91125

## INTRODUCTION

To us, and to other biological organisms, vision seems effortless. We open our eyes and we "see" the world in all its color, brightness, and movement. Yet, we have great difficulties when trying to endow our machines with similar abilities. In this paper we shall describe recent developments in the theory of early vision which lead from the formulation of the motion problem as an ill-posed one to its solution by minimizing certain "cost" functions. These cost or energy functions can be mapped onto simple analog and digital resistive networks. Thus, we shall see how the optical flow can be computed by injecting currents into resistive networks and recording the resulting stationary voltage distribution at each node. These networks can be implemented in cMOS VLSI circuits and represent plausible candidates for biological vision systems.

## APERTURE PROBLEM AND SMOOTHNESS ASSUMPTION

In this study, we use intensity-based schemes for recovering motion. Let us derive an equation relating the change in image brightness to the motion of the image (see[1]). Let us assume that the brightness of the image is constant over time: $dI(x, y, t)/dt = 0$. On the basis of the chain rule of differentiation, this transforms into

$$\frac{\partial I}{\partial x}\frac{dx}{dt} + \frac{\partial I}{\partial y}\frac{dy}{dt} + \frac{\partial I}{\partial t} = I_x u + I_y v + I_t = \nabla I \cdot v + I_t = 0, \qquad (1)$$

where we define the velocity v as $(u, v) = (dx/dt, dy/dt)$. Because we assume that we can compute these spatial and temporal image gradients, we are now left with a single linear equation in two unknowns, $u$ and $v$, the two components of the velocity vector (aperture problem). Any measuring system with a finite aperture, whether biological or artificial, can only sense the velocity component perpendicular to the edge or along the spatial gradient $(-I_t/ \mid \nabla I \mid)$. The component of motion perpendicular to the gradient cannot, in principle, be registered. The problem remains unchanged even if we measure these velocity components at many points throughout the image.

How can this problem be made well-posed, that is, having a unique solution depending continuously on the data? One form of "regularizing" ill-posed

problems is to restrict the class of admissible solutions by imposing appropriate constraints[2]. Applying this method to motion, we shall argue that in general objects are smooth—except at isolated discontinuities—undergoing smooth movements. Thus, in general, neighboring points in the world will have similar velocities and the projected velocity field should reflect this fact. We therefore impose on the velocity field the constraint that it should be the smoothest as well as satisfying the data. As measure of smoothness we choose, the square of the velocity field gradient. The final velocity field $(u, v)$ is the one that minimizes

$$E(u, v) = \iint (I_x u + I_y v + I_t)^2 +$$

$$\lambda \iint \left[ \left( \frac{\partial u}{\partial x} \right)^2 + \left( \frac{\partial u}{\partial y} \right)^2 + \left( \frac{\partial v}{\partial x} \right)^2 + \left( \frac{\partial v}{\partial y} \right)^2 \right] dx \, dy \qquad (2)$$

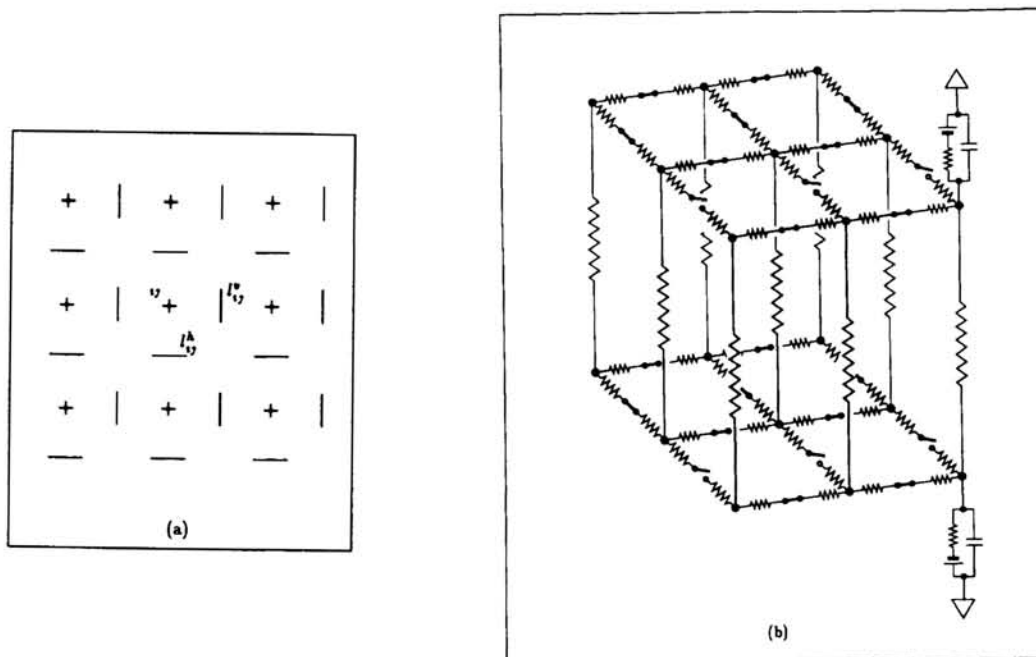

(a)

(b)

Fig. 1. (a) The location of the horizontal $(l_{ij}^h)$ and vertical $(l_{ij}^v)$ line processes relative to the motion field nngrid. (b) The hybrid resistive network, computing the optical flow in the presence of discontinuities. The conductances $T_{c-ij}$ connecting both grids depend on the brightness gradient, as do the conductances $g_{ij}^u$ and $g_{ij}^v$ connecting each node with the battery. For clarity, only two such elements are shown. The battery $E_{ij}$ depends on both the temporal and the spatial gradient and is zero if no brightness change occurs. The $x$ (resp. $y$) component of the velocity is given by the voltage in the top (resp. bottom) network. Binary switches, which make or break the resistive connections between nodes,

implement motion discontinuities. These switches could be under the control of distributed digital processors. Analog cMOS implementations are also feasible[3].

The first term implements the constraint that the final solution should follow as closely as possible the measured data whereas the second term imposes the smoothness constraint on the solution. The degree to which one or the other terms are minimized is governed by the parameter $\lambda$. If the data is very accurate, it should be "expensive" to violate the first term and $\lambda$ will be small. If, conversely, the data is unreliable (low signal-to-noise), much more emphasis will be placed on the smoothness term. Horn and Schunck[1] first formulated this variational approach to the motion problem.

The energy $E(u, v)$ is quadratic in the unknown $u$ and $v$. It then follows from standard calculus of variation that the associated Euler-Lagrange equations will be linear in $u$ and $v$:

$$
\begin{aligned}
I_x^2 u + I_x I_y v - \lambda \nabla^2 u + I_x I_t = 0 \\
I_x I_y u + I_y^2 v - \lambda \nabla^2 v + I_y I_t = 0.
\end{aligned}
\tag{3}
$$

We now have two linear equations at every point and our problem is therefore completely determined.

## ANALOG RESISTIVE NETWORKS

Let us assume that we are formulating eqs. (2) and (3) on a discrete 2-D grid, such as the one shown in fig. 1a. Equation (3) then transforms into

$$
\begin{aligned}
I_{xij}^2 u_{ij} + I_{xij} I_{yij} v_{ij} - \lambda \left( u_{i+1j} + u_{ij+1} - 4u_{ij} + u_{i-1j} + u_{ij-1} \right) + I_{xij} I_{tij} = 0 \\
I_{xij} I_{yij} u_{ij} + I_{yij}^2 v_{ij} - \lambda \left( v_{i+1j} + v_{ij+1} - 4v_{ij} + v_{i-1j} + v_{ij-1} \right) + I_{yij} I_{tij} = 0
\end{aligned}
\tag{4}
$$

where we replaced the Laplacian with its 5 point approximation on a rectangular grid. We shall now show that this set of linear equations can be solved naturally using a particular simple resistive network. Let us apply Kirchhoff's current law to the nodne $i, j$ in the top layer of the resistive network shown in fig. 1b. We then have the following update equation:

$$
\begin{aligned}
C \frac{du_{ij}}{dt} = {} & T \left( u_{i+1j} + u_{ij+1} - 4u_{ij} + u_{i-1j} + u_{ij-1} \right) \\
& + g_{ij}^u (E_{ij} - u_{ij}) + T_{c-ij}(v_{ij} - u_{ij}).
\end{aligned}
\tag{5}
$$

where $v_{ij}$ is the voltage at node $i, j$ in the bottom network. Once $du_{ij}/dt = 0$ and $dv_{ij}/dt = 0$, this equation is seen to be identical with eq. (4), if we identify

$$T \longrightarrow \lambda$$
$$T_{c-ij} \longrightarrow -I_{xij}I_{yij}$$
$$g_{ij}^u \longrightarrow I_{xij}(I_{xij} + I_{yij})$$
$$g_{ij}^v \longrightarrow I_{yij}(I_{xij} + I_{yij})$$
$$E_{ij} \longrightarrow \frac{-I_t}{I_{xij} + I_{yij}}.$$

$$(6)$$

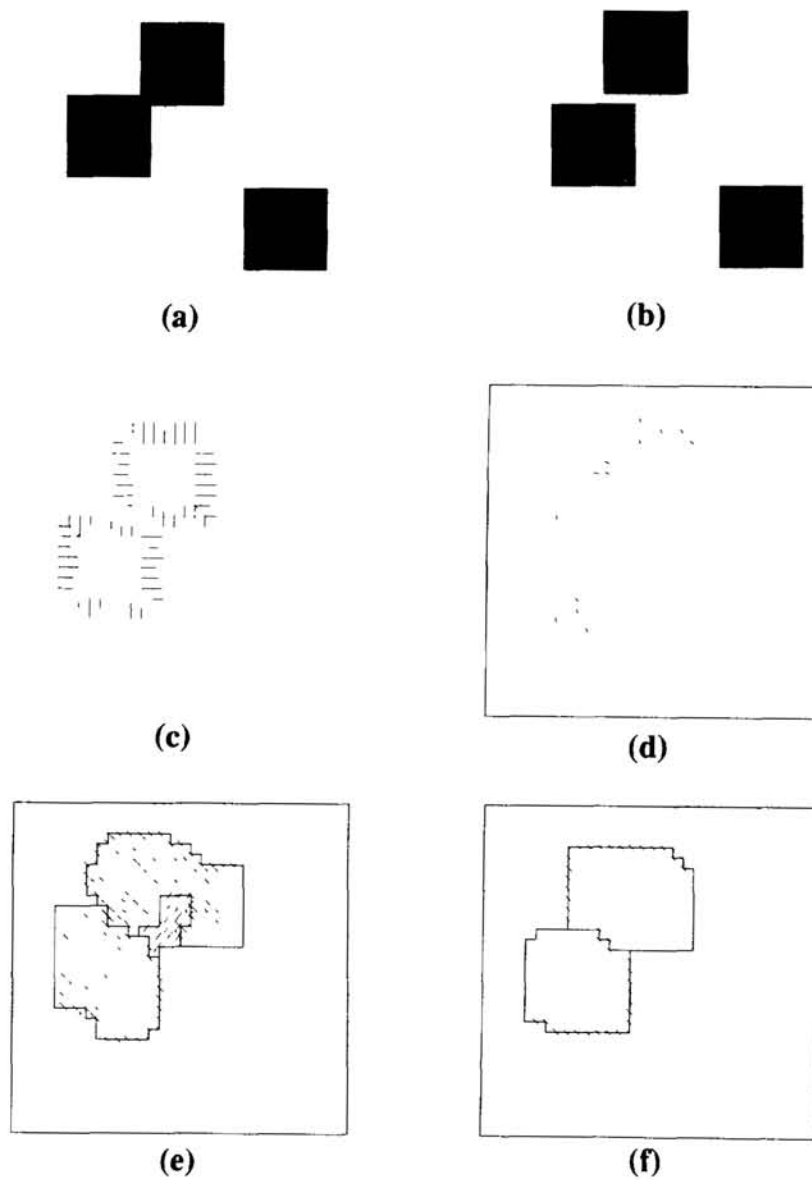

(a)　　　　　　　　(b)

(c)　　　　　　　　(d)

(e)　　　　　　　　(f)

Fig. 2. Motion sequence using synthetic data. (a) and (b)  Two images of three high contrast squares on a homogeneous background. (c)  The initial velocity data. The inside of both squares contain no data. (d)  The final state

of the network after 240 iterations, corresponding to the smooth optical flow field. (e) Optical flow in the presence of motion discontinuities (indicated by solid lines). (f) Discontinuities are strongly encouraged to form at the location of intensity edges[4]. Both (e) and (f) show the state of the hybrid network after six analog-digital cycles.

Once we set the batteries and the conductances to the values indicated in eq. (6), the network will settle—following Kirchhoff's laws—into the state of least power dissipation. The associated stationary voltages correspond to the sought solution: $u_{ij}$ is equivalent to the $x$ component and $v_{ij}$ to the $y$ component of the optical flow field.

We simulated the behavior of these networks by solving the above circuit equations on parallel computers of the Hypercube family. As boundary conditions we copied the initial velocity data at the edge of the image into the nodes lying directly adjacent but outside the image.

The sequences in figs. 2 and 3 illustrate the resulting optical flow for synthetic and natural images. As discussed by Horn and Schunck[1], the smoothness constraint leads to a qualitatively correct estimate of the velocity field. Thus, one undifferentiated blob appears to move to the lower right and one blob to the upper left. However, at the occluding edge where both squares overlap, the smoothness assumption results in a spatial average of the two opposing velocities, and the estimated velocity is very small or zero. In parts of the image where the brightness gradient is zero and thus no initial velocity data exists (for instance, the interiors of the two squares), the velocity estimates are simply the spatial average of the neighboring velocity estimates. These empty areas will eventually fill in from the boundary, similar to the flow of heat for a uniform flat plate with "hot" boundaries.

## MOTION DISCONTINUITIES

The smoothness assumption of Horn and Schunck[1] regularizes the aperture problem and leads to the qualitatively correct velocity field inside moving objects. However, this approach fails to detect the locations at which the velocity changes abruptly or discontinuously. Thus, it smoothes over the figure-ground discontinuity or completely fails to detect the boundary between two objects with differing velocities because the algorithm combines velocity information across motion boundaries.

A quite successful strategy for dealing with discontinuities was proposed by Geman and Geman[5]. We shall not rigorously develop their approach, which is based on Bayesian estimation theory (for details see[5,6]). Suffice it to say that a priori knowledge, for instance, that the velocity field should in general be smooth, can be formulated in terms of a Markov Random Field model of the image. Given such an image model, and given noisy data, we then estimate the "best" flow field by some likelihood criterion. The one we will use here

is the maximum a posteriori estimate, although other criteria are possible and have certain advantages[6]. This can be shown to be equivalent to minimizing an expression such as eq. (2).

In order to reconstruct images consisting of piecewise constant segments, Geman and Geman[5] further introduced the powerful idea of a line process $l$. For our purposes, we will assume that a line process can be in either one of two states: "on" ($l = 1$) or "off" ($l = 0$). They are located on a regular lattice set between the original pixel lattice (see fig. 1a), such that each pixel $i,j$ has a horizontal $l_{ij}^h$ and a vertical $l_{ij}^v$ line process associated with it. If the appropriate line process is turned on, the smoothness term between the two adjacent pixels will be set to zero. In order to prevent line processes from forming everywhere and, furthermore, in order to incorporate additional knowledge regarding discontinuities into the line processes, we must include an additional term $V_c(l)$ into the new energy function:

$$E(u,v,l^h,l^v) = \sum_{i,j} \left(I_x u_{ij} + I_y v_{ij} + I_t\right)^2 +$$

$$\lambda \sum_{i,j} \left(1 - l_{ij}^h\right) \left[(u_{i+1j} - u_{ij})^2 + (v_{i+1j} - v_{ij})^2\right] + \qquad (7)$$

$$\lambda \sum_{i,j} \left(1 - l_{ij}^v\right) \left[(u_{ij+1} - u_{ij})^2 + (v_{ij+1} - v_{ij})^2\right] + V_c(l).$$

$V_c$ contains a number of different terms, penalizing or encouraging specific configurations of line processes:

$$V_c(l) = C_c \sum_{i,j} l_{ij}^h + C_p \sum_{i,j} l_{ij}^h \left(l_{ij+1}^h + l_{ij+2}^h\right) + C_I V_I(1), \qquad (8)$$

plus the corresponding expression for the vertical line process $l_{ij}^v$ (obtained by interchanging $i$ with $j$ and $l_{ij}^v$ with $l_{ij}^h$). The first term penalizes each introduction of a line process, since the cost $C_c$ has to be "payed" every time a line process is turned on. The second term prevents the formation of parallel lines: if either $l_{ij+1}^h$ or $l_{ij+2}^h$ is turned on, this term will tend to prevent $l_{ij}^h$ from turning on. The third term, $C_I V_I$, embodies the fact that in general, motion discontinuities occur along extended contours and rarely intersect (for more details see[7]).

We obtain the optical flow by minimizing the cost function in eq. (7) with respect to both the velocity $v$ and the line processes $l^h$ and $l^v$. To find an optimal solution to this non-quadratic minimization problem, we follow Koch et al.[7] and use a purely deterministic algorithm, based on solving Kirchhoff's equations for a mixed analog/digital network (see also [8]). Our algorithm exploits the fact that for a fixed distribution of line processes, the energy function (7) is quadratic. Thus, we first initialize the analog resistive network (see fig. 2b) according to eq. (6) and with no line processes on. The network then converges to the smoothest solution. Subsequently, we update the line processes by deciding at each site of the line process lattice whether the overall energy can be lowered by setting or breaking the line process; that is, $l_{ij}^h$ will be turned on if $E(u, v, l_{ij}^h = l, l^v) < E(u, v, l_{ij}^h = 0, l^v)$; otherwise, $l_{ij}^h = 0$. Line processes are switched on by breaking the appropriate resistive connection between the two neighboring nodes. After the completion of one such analog-digital cycle, we reiterate and compute—for the newly updated distribution of line processes—the smoothest state of the analog network. Although there is no guarantee that the system will converge to the global minimum, since we are using a gradient descent rule, it seems to find next-to-optimal solutions in about 10 to 15 analog-digital cycles.

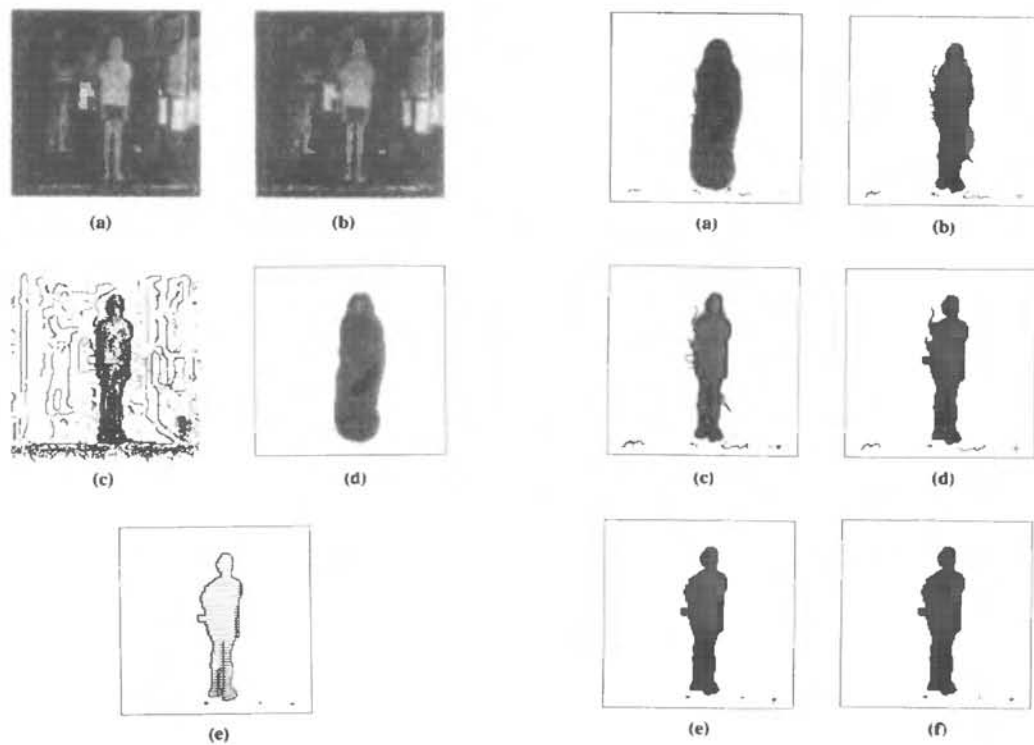

Figure 3. Optical flow of a moving person. (a) and (b) Two 128 by 128 pixel images captured by a video camera. The person in the foreground is moving toward the right while the person in the background is stationary. The noise in the lower part of the image is a camera artifact. (c) Zero-crossings superimposed on the initial velocity data. (d) The smooth optical flow after 1000 iterations. Note that the noise in the lower part of both images is completely smoothed away. (e) The final piecewise smooth optical flow. The velocity field is subsampled to improve visibility. The evolution of the hybrid network is shown after the 1. (a), 3. (b), 5. (c), 7. (d), 10. (e), and 13. (f) analog-digital cycle in the right part of the figure.

The synthetic motion sequence in fig. 2 demonstrates the effect of the line

processes. The optical flow outside the discontinuities approximately delineating the boundaries of the moving squares is zero, as it should be (fig. 2e). However, where the two squares overlap the velocity gradient is high and multiple intersecting discontinuities exist. To restrict further the location of discontinuities, we adopt a technique used by Gamble and Poggio[4] to locate depth discontinuities by requiring that depth discontinuities coincide with the location of intensity edges. Our rationale behind this additional constraint is that with very few exceptions, the physical processes and the geometry of the 3-dimensional scene giving rise to the motion discontinuity will also give rise to an intensity edge. As edges we use the zero-crossings of a Laplacian of a Gaussian convolved with the original image[9]. We now add a new term $V_{Z-C_{ij}}$ to our energy function $E$, such that $V_{Z-C_{ij}}$ is zero if $l_{ij}$ is off or if $l_{ij}$ is on and a zero-crossing exists between locations $i$ and $j$. If $l_{ij} = 1$ in the absence of a zero-crossing, $V_{Z-C_{ij}}$ is set to 1000. This strategy effectively prevents motion discontinuities from forming at locations where no zero-crossings exist, unless the data strongly suggest it. Conversely, however, zero-crossings by themselves will not induce the formation of discontinuities in the absence of motion gradients (figs. 2f and 3).

## ANALOG VLSI NETWORKS

Even with the approximations and optimizations described above, the computations involved in this and similar early vision tasks require minutes to hours on computers. It is fortunate then that modern integrated circuit technology gives us a medium in which extremely complex, analog real-time implementations of these computational metaphors can be realized[3].

We can achieve a very compact implementation of a resistive network using an ordinary cMOS process, provided the transistors are run in the sub-threshold range where their characterstics are ideal for implementing low-current analog functions. The effect of a resistor is achieved by a circuit configuration, such as the one shown in fig. 4, rather than by using the resistance of a special layer in the process. The value of the resulting resistance can be controlled over three orders of magnitude by setting the bias voltages on the upper and lower current source transistors. The current-voltage curve saturates above about 100 mV; a feature that can be used to advantage in many applications. When the voltage gradients are small, we can treat the circuit just as if it were a linear resistor. Resistances with an effective negative resistance value can easily be realized.

In two dimensions, the ideal configuration for a network implementation is shown in fig. 4. Each point on the hexagonal grid is coupled to six equivalent neighbors. Each node includes the resistor apparatus, and a set of sample-and-hold circuits for setting the confidence and signal the input and output voltages. Both the sample-and-hold circuits and the output buffer are addressed by a scanning mechanism, so the stored variables can be refreshed or updated, and the map of node voltages read out in real time.

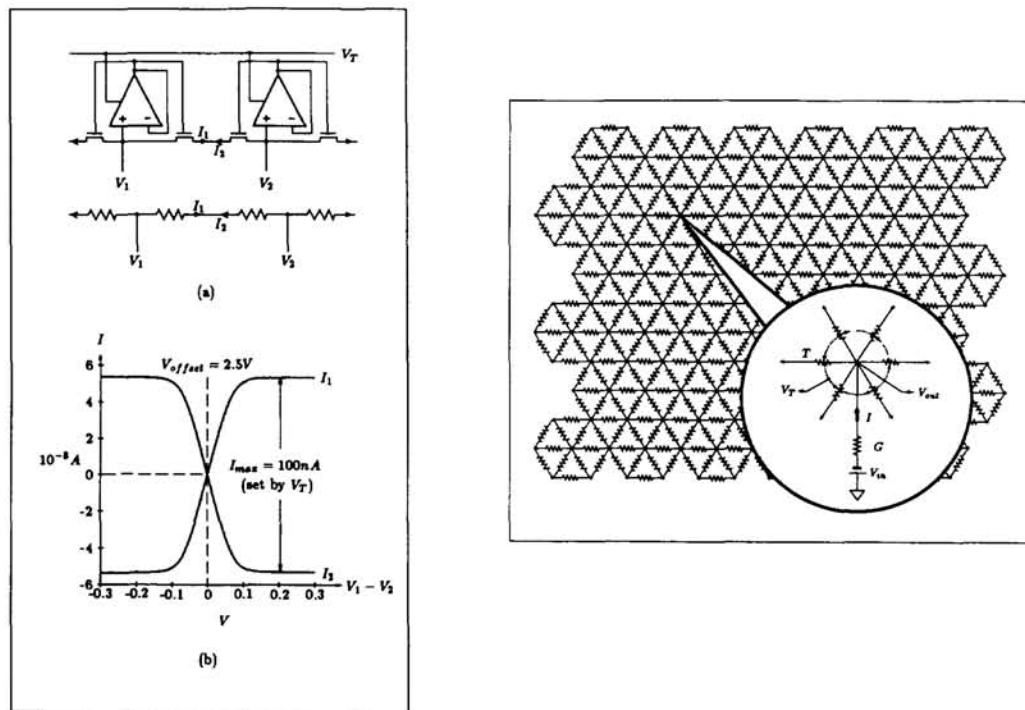

Figure 4. Circuit design for a resistive network for interpolating and smoothing noisy and sparsely sampled depth measurements. (a) Circuit—consisting of 8 transistors—implementing a variable nonlinear resistance. (b) If the voltage gradient is below 100 mV its approximates a linear resistance. The voltage $V_T$ controls the maximum current and thus the slope of the resistance, which can vary between 1 $M\Omega$ and 1 $G\Omega$ [3]. This cMOS circuit contains 20 by 20 grid points on a hexagonal lattice. The individual resistive elements with a variable slope controlled by $V_T$ correspond to the term governing the smoothness, $\lambda$. At those locations where a depth measurement $d_{ij}$ is present, the battery is set to this value ($V_{in} = d_{ij}$) and the value of the conductance $G$ is set to some fixed value. If no depth data is present at that node, $G$ is set to zero. The voltage at each node corresponds to the discrete values of the smoothed surface fitted through the noisy and sparse measurements[7].

A 48 by 48 silicon retina has been constructed that uses the hexagonal network of fig. 4 as a model for the horizontal cell layer in the vertebrate retina[10]. In this application, the input potentials were the outputs of logarithmic photoreceptors—implemented via phototransistors—and the potential difference across the conductance $T$ formed an excellent approximation to the Laplacian operator.

## DISCUSSION

We have demonstrated in this study that the introduction of binary motion

discontinuities into the algorithm of Horn and Schunck[1] leads to a dramatically improved performance of their method, in particular for the optical flow in the presence of a number of moving non-rigid objects. Moreover, we have shown that the appropriate computations map onto simple resistive networks. We are now implementing these resistive networks into VLSI circuits, using subtheshold cMOS technology. This approach is of general interest, because a great number of problems in early vision can be formulated in terms of similar non-convex energy functions that need to be minimized, such as binocular stereo, edge detection, surface interpolation, structure from motion, etc.[2,6,8].

These networks share several features with biological neural networks. Specifically, they do not require a system-wide clock, they rely on many connections between simple computational nodes, they converge rapidly—within several time constants—and they are quite robust to hardware errors. Another interesting feature is that our networks only consume very moderate amounts of power; the entire retina chip requires about 100 $\mu$W [10]

Acknowledgments: An early version of this model was developed and implemented in collaboration with A. L. Yuille[8]. M. Avalos and A. Hsu wrote the code for the Imaging Technology system and E. Staats for the NCUBE. C.K. is supported by an ONR Research Young Investigator Award and by the Sloan and the Powell Foundations. C.M. is supported by ONR and by the System Development Foundation. A portion of this research was carried out at the Jet Propulsion Laboratory and was sponsored by NSF grant No. EET-8714710, and by NASA.

## REFERENCES

1. Horn, B. K. P. and Schunck, B. G. Artif. Intell. 17, 185–203 (1981).
2. Poggio, T., Torre, V. and Koch, C. Nature 317, 314–319 (1985).
3. Mead, C. Analog VLSI and Neural Systems. Addison-Wesley: Reading, MA (1988).
4. Gamble, E. and Poggio, T. Artif. Intell. Lab. Memo. No. 970, MIT, Cambridge MA (1987).
5. Geman, S. and Geman, D. IEEE Trans. PAMI 6, 721–741 (1984).
6. Marroquin, J., Mitter, S. and Poggio, T. J. Am. Stat. Assoc. 82, 76–89 (1987).
7. Koch, C., Marroquin, J. and Yuille, A. Proc. Natl. Acad. Sci. USA 83, 4263–4267 (1986).
8. Yuille, A. L. Artif. Intell. Lab. Memo. No. 987, MIT, Cambridge, MA (1987).
9. Marr, D. and Hildreth, E. C. Proc. R. Soc. Lond. B 207, 187–217 (1980).
10. Sivilotti, M. A., Mahowald, M. A. and Mead, C. A. In: 1987 Stanford VLSI Conference, ed. P. Losleben, pp. 295–312 (1987).
